# Analytic Solutions to the Formation of Feature-Analysing Cells of a Three-Layer Feedforward Visual Information Processing Neural Net

D.S. Tang
Microelectronics and Computer Technology Corporation
3500 West Balcones Center Drive
Austin, TX  78759-6509
email: tang@mcc.com

## ABSTRACT

Analytic solutions to the information-theoretic evolution equation of the connection strength of a three-layer feedforward neural net for visual information processing are presented. The results are (1) the receptive fields of the feature-analysing cells correspond to the eigenvector of the maximum eigenvalue of the Fredholm integral equation of the first kind derived from the evolution equation of the connection strength; (2) a symmetry-breaking mechanism (parity-violation) has been identified to be responsible for the changes of the morphology of the receptive field; (3) the conditions for the formation of different morphologies are explicitly identified.

## 1 INTRODUCTION

The use of Shannon's information theory ( Shannon and Weaver,1949) to the study of neural nets has been shown to be very instructive in explaining the formation of different receptive fields in the early visual information processing, as evident by the works of Linsker (1986,1988). It has been demonstrated that the connection strengths which maximize the information rate from one layer of neurons to the next exhibit center-surround, all-excitatory/all-inhibitory and orientation-selective properties. This could lead to a better understanding on the mechanisms with which the cells are self-organized to achieve adaptive responses to the changing enviroment. However, results from these studies are mainly numerical in nature and therefore do not provide deeper insights as to how and under what conditions the morphologies of the feature-analyzing cells are formed. We present in this paper

accurate analytic solutions to the problems posed by Linsker. Namely, we solve analytically the evolution equation of the connection strength, obtain close expressions for the receptive fields and derive the formation conditions for different classes of morphologies. These results are crucial to the understanding of the architecture of neural net as an information processing system. Below, we briefly summarize the analytic techniques involved and the main results we obtained.

## 2 THREE-LAYER FEEDFORWARD NEURAL NET

The neural net configuration (Fig. 1) is identical to that reported in references 2 and 3 in which a feedforword three-layer neural net is considered. The layers are labelled consecutively as layer-A, layer-B and layer-C.

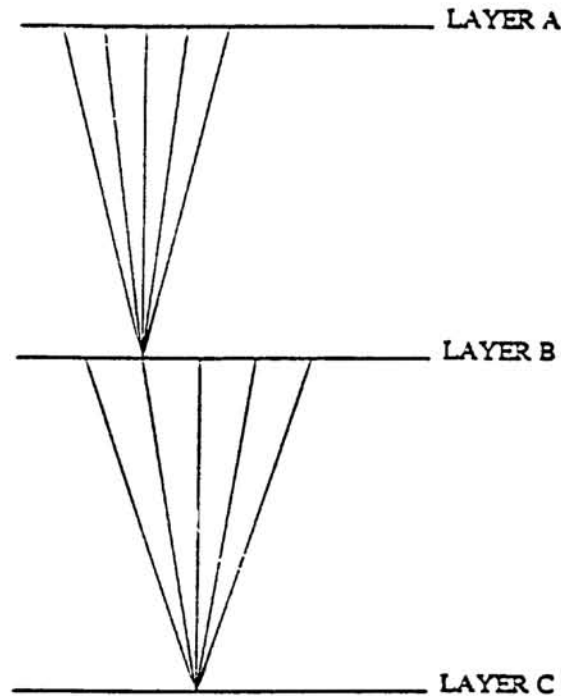

Figure 1: The neural net configuration

The input-output relation for the signals to propagate from one layer to the consecutive layer is assumed to be linear,

$$M_j = \sum_{i=1}^{N_j} C_{ji}(L_i + n_i). \tag{1}$$

$n_i$ is assumed to be an additive Gaussian white noise with constant standard deviation $\alpha$ and zero mean. $L_i$ and $M_j$ are the ith stochastic input signal and the jth stochastic output signal respectively. $C_{ji}$ is the connection strength which defines the morphology of the receptive field and is to be determined by maximizing the information rate. The spatial summation in equation (1) is to sum over all $N_j$

inputs located according to a gaussian distributed within the same layer, with the center of the distribution lying directly above the location of the $M_j$ output signal. If the statistical behavior of the input signal is assumed to be Gaussian,

$$P(L) = \frac{e^{-\sum_{ij} L_i Q_{ij}^{-1} L_j}}{(2\pi)^{\frac{N}{2}} \sqrt{Det(Q)}}, \tag{2}$$

then the information rate can be derived and is given by

$$R(M) = \frac{1}{2} ln[1 + \frac{\sum C_i Q_{ij} C_j}{\alpha^2 \sum C_i^2}]. \tag{3}$$

The matrix $Q$ is the correlation of the L's, $Q_{ij} = E[(L_i - \tilde{L})(L_j - \tilde{L})]$ with mean $\tilde{L}$. The set of connection strengths which optimize the information rate subject to a normalization condition, $\sum C_i^2 = A$, and to their overall absolute mean, $(\sum C_i)^2 = B$, constitute physically plausible receptive fields. Below is the solutions to the problem.

## 3 FREDHOLM INTEGRAL EQUATION

The evolution equation for the connection strength $C_n$ which maximizes the information rate subject to the constraints is

$$\dot{C}_n = \frac{1}{N} \sum_{i=1}^{N} (Q_{ni} + k_2) C_i. \tag{4}$$

$k_2$ is the Lagrange multiplier. First, we assume that the statistical ensemble of the visual images has the highest information content under the condition of fixed variance. Then, from the maximum entropy principle, it can be shown that the Gaussian distribution with a correlation $Q_{ij}$ being a constant multiple of the kronecker delta function describes the statistics of this ensemble of visual images. It can be shown that the solution to the above equation with $Q_{ni}$ being a kronecker delta function is a constant. Therefore, the connection strengths which defines the linear input-output relation from layer A to layer B is either all-excitatory or all-inhibitory. Hence, without loss of generality, we take the values of the layer A to layer B connection strengths to be all-excitatory. Making use of this result, the correlation function of the output signals at layer B (i.e. the input signals to layer C) is derived

$$Q_{ni} = C_Q exp(-r^2/2r_B^2) \tag{5}$$

where r is the distance between the nth and the ith output signals. $C_Q = \pi N/$ 50. To study the connection strengths of the input-output relation from layer B to layer C, it is more convenient to work with continuous spatial variables. Then the solutions to the discrete evolution equation which maximizes the information rate are solutions to the following Fredholm integral equation of the first kind with the maximum eigenvalue $\lambda$,

$$C(\vec{r}) = \frac{1}{N\lambda} \int_{-\infty}^{+\infty} K(\vec{R}|\vec{r}) C(\vec{R}) d\vec{R} \tag{6}$$

where the kernal is $K(\vec{R}|\vec{r}) = (Q(\vec{R}-\vec{r})+k_2)\rho(\vec{R})$ and the Gaussian input population distribution density is $\rho(\vec{r}) = C_\rho exp(-\frac{r^2}{r_B^2})$ with $C_\rho = \frac{N}{\pi r_B^2}$. In continuous variables, the connection strength is denoted by $C(\vec{r})$. A complete set of solutions to this Fredholm integral equation can be analytically derived. We are interested only in the solutions with the maximum eigenvalues. Below we present the results.

## 4 ANALYTIC SOLUTIONS

The solution with the maximum eigenvalue has a few number of nodes. This can be constructed as a linear superposition of an infinite number of gaussian functions with different variances and means, which are treated as independent variables to be solved with the Fredholm integral equation. Full details are contained in reference 3.

(a) Symmetric solution $C(-\vec{r}) = C(\vec{r})$:
For $k_2 \neq 0$, the connection strength is

$$C(\vec{r}) = b[1 + Gexp(-\frac{r^2}{2\sigma_0^2}) + \frac{H}{(1-H)}Gexp(-\frac{r^2}{2\sigma_\infty^2})] \tag{7}$$

with $G = \frac{a\pi}{\frac{1}{2r_B^2}+\frac{1}{2a^2}}$ and $H = \frac{a\pi}{\frac{1}{2r_B^2}+\frac{1}{2a^2}+\frac{1}{2\sigma_0^2}}$. Here, $\alpha^2 = .5r_B^2$, $\frac{r_B^2}{\sigma_0^2} = 0.66667$, $\frac{r_B^2}{\sigma_\infty^2} = 0.73205$ and $a \equiv C_Q C_\rho/N\lambda$.

The eigenvalue is given by

$$\lambda = \frac{k_2 C_\rho \pi}{N}[2\alpha^2 + \frac{G}{\frac{1}{2\sigma_0^2}+\frac{1}{2\alpha^2}} + \frac{H}{(1-H)}G\frac{1}{\frac{1}{2\sigma_\infty^2}+\frac{1}{2\alpha^2}}]. \tag{8}$$

For $k_2 = 0$, the connection strength is

$$C(\vec{r}) = fexp(-\frac{r^2}{2\sigma^2}) \tag{9}$$

and the eigenvalue is

$$\lambda = \frac{C_Q C_\rho \pi}{N[\frac{1}{2r_B^2} + \frac{1}{2\alpha^2} + \frac{1}{2\sigma_\infty^2}]}. \tag{10}$$

These can be shown to be identical to the case of $k_2 \neq 0$ when the limit $k_2 \to 0$ is appropriately taken.

(b) Antisymmetric solution $C(-\vec{r}) = -C(\vec{r})$:
The connection strength is

$$C(\vec{r}) = (fx + gy)exp(-\frac{r^2}{2r_B^2}[1 - \frac{1}{1+\frac{r_B^2}{\alpha^2}+\frac{r_B^2}{\sigma_\infty^2}}]). \tag{11}$$

The eigenvalue is

$$\lambda = \frac{\pi C_Q C_\rho}{N2r_B^2[\frac{1}{2r_B^2} + \frac{1}{2\alpha^2} + \frac{1}{2\sigma_\infty^2}]^2}. \tag{12}$$

In the above equations, $b$, $f$ and $g$ are normalization constants.

Below are the conditions under which the different morphologies (Fig.2 ) are formed.

(i)$k_2 > 0$, the symmetric solution has the largest eigenvalue. The receptive field is either all-excitatory or all-inhibitory, Fig.2a.

(ii)$-0.891C_Q < k_2 < 0$, the symmetric solution has the largest eigenvalue. The receptive field has a mexcian-hat appearance, Fig.2b.

(iii)$k_2 < -0.891C_Q$, the anti-symmetric solution has the largest eigenvalue. The receptive field has two regions divided by a straight line of arbitrary direction(degeneracy). The two regions are mirror image of each other. One is totally inhibitory and the other is totally excitatory, Fig.2c.

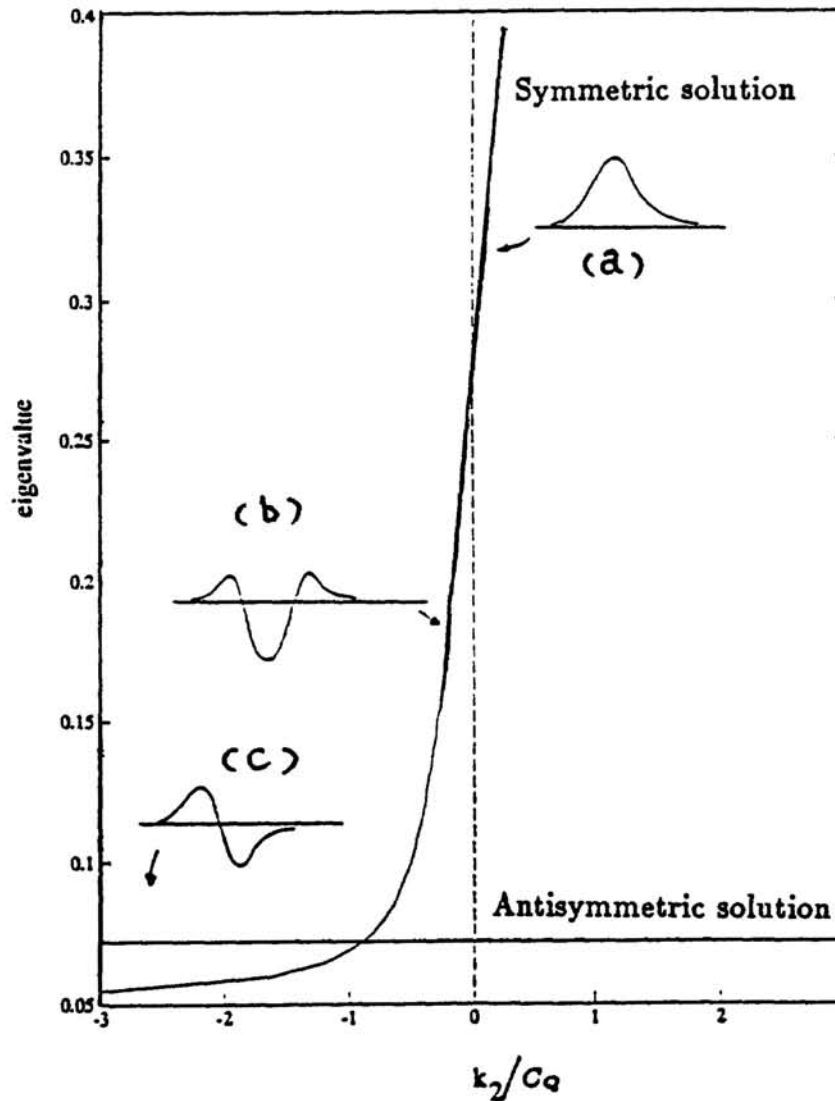

Figure 2: Relations between the receptive field and the maximum eigenvalues. Inserts are examples of the connection strength $C(\vec{r})$ versus the spatial dimension in the x-direction.

Note that the information rate as given by Eq.(3) is invariant under the operation of the spatial reflection,$-\vec{r} \rightarrow \vec{r}$. The solutions to the optimaziation problem violates parity-conservation as the overall mean of the connection strength (i.e. equivalently $k_2$) changes to different values.

Results from numerical simulations agree very well with the analytic results. Numerical simulations are performed from 80 to 600 synapses. The agreement is good even for the case in which the number of synapses are 200.

In summary, we have shown precisely how the mexican-hat morphology emerges as identified by (ii) above. Furthermore, a symmetry-breaking(parity-violation) mechanism has been identified to explain the changes of the morphology from spatially symmetric to anti-symmetric appearance as $k_2$ passes through $-0.891C_Q$. It is very likely that similar symmetry breaking mechanisms are present in neural nets with lateral connections.

## References

1. C.E.Shannon and W. Weaver, The mathematical Theory of Communication (Univ. of Illinois Press,Urbana,1949).

2. R.Linsker, Proc. Natl. Acad. Sci. USA 83,7508(1986); Computer 21 (3), 105(1988).

3. D.S. Tang, Phys.Rev A, 40,6626(1989).
